# Dynamical Modeling with Kernels for Nonlinear Time Series Prediction

**Liva Ralaivola**
Laboratoire d'Informatique de Paris 6
Université Pierre et Marie Curie
8, rue du capitaine Scott
F-75015 Paris, FRANCE
liva.ralaivola@lip6.fr

**Florence d'Alché–Buc**
Laboratoire d'Informatique de Paris 6
Université Pierre et Marie Curie
8, rue du capitaine Scott
F-75015 Paris, FRANCE
florence.dalche@lip6.fr

## Abstract

We consider the question of predicting nonlinear time series. Kernel Dynamical Modeling (KDM), a new method based on kernels, is proposed as an extension to linear dynamical models. The kernel trick is used twice: first, to learn the parameters of the model, and second, to compute preimages of the time series predicted in the feature space by means of Support Vector Regression. Our model shows strong connection with the classic Kalman Filter model, with the kernel feature space as hidden state space. Kernel Dynamical Modeling is tested against two benchmark time series and achieves high quality predictions.

## 1 Introduction

Prediction, smoothing and filtering are traditional tasks applied to time series. The machine learning community has recently paid a lot of attention to these problems and especially to nonlinear time series prediction in various areas such as biological signals, speech or financial markets. To cope with non linearities, extensions of the Kalman filter [5, 4] have been proposed for filtering and smoothing while recurrent artificial neural networks [2] and support vector regressors [7, 8] have been developed for prediction purposes. In this paper, we focus on prediction tasks and introduce a powerful method based on the *kernel trick* [1], which has been successfully used in tasks ranging from classification and regression to data analysis (see [13, 15] for details). Time series modeling is addressed by extending the framework of observable linear dynamical systems [12] to the *feature space* defined by a kernel. The predictions are realized in the feature space and are then transformed to obtain the corresponding *preimages* in the input space. While the proposed model could be used for smoothing as well as filtering, we here focus on the prediction task. A link to the Kalman filter can be drawn by noticing that given the efficiency of our model for the prediction task it can be used as a hidden transition process in the Kalman filter setting.

The paper is organized as follows. In the next section, we describe how the modeling of a time series can take place in the feature space and explain how to solve the *preimage problem* by a learning strategy. In the third section, we present prediction results achieved by our model In the fourth section, the estimation algorithm is discussed and its link to the Kalman filter is highlighted. We finally conclude by giving some perspectives to our work.

## 2  Principles of Dynamical Modeling with Kernels

### 2.1  Basic Formulation

The problem we address is that of modeling $d$-dimensional nonlinear real-valued time series defined as

$$\mathbf{x}_{t+1} = h(\mathbf{x}_t) + \mathbf{u} \tag{1}$$

from an observed sequence $\mathbf{x}_{1:T} = \{\mathbf{x}_1, \ldots, \mathbf{x}_T\}$ produced by this model, where $h$ is a (possibly unknown) nonlinear function and $\mathbf{u}$ a noise vector.

Modeling such a series can be done with the help of recurrent neural networks [2] or support vector machines [7]. In this work, we instead propose to deal with this problem by extending linear dynamical modeling thanks to the kernel trick. Instead of considering the observation sequence $\mathbf{x}_{1:T} = \{\mathbf{x}_1, \ldots, \mathbf{x}_T\}$, we consider the sequence $\mathbf{x}_{1:T}^\phi = \{\phi(\mathbf{x}_1), \ldots, \phi(\mathbf{x}_T)\}$, where $\phi$ is a mapping from $\mathbb{R}^d$ to $\mathcal{H}$ and $k$ its associated kernel function [15] such that $k(\mathbf{v}_1, \mathbf{v}_2) = \langle \phi(\mathbf{v}_1), \phi(\mathbf{v}_2) \rangle \ \forall \mathbf{v}_1, \mathbf{v}_2 \in \mathbb{R}^d$, $\langle \cdot, \cdot \rangle$ being the inner product of $\mathcal{H}$. The *Kernel Dynamical Model* (KDM) obtained can be written as:

$$\mathbf{x}_{t+1}^\phi = A^\phi \mathbf{x}_t^\phi + \boldsymbol{\mu}^\phi + \boldsymbol{\nu}^\phi \tag{2}$$

where $A^\phi$ is the process transition matrix, $\boldsymbol{\mu}^\phi$ an offset vector, $\boldsymbol{\nu}^\phi \in \mathcal{H}$ a gaussian isotropic noise of magnitude $\sigma^2$ and $\mathbf{x}_t^\phi$ stands for $\phi(\mathbf{x}_t)$.

We are going to show that it is possible to apply the maximum likelihood principle to identify $\sigma^2$, $A^\phi$ and $\boldsymbol{\mu}^\phi$ and come back to the input space thanks to preimages determination.

### 2.2  Estimation of the Model Parameters

Learning the parameters of the model (2) by maximum likelihood given an observation sequence $\mathbf{x}_{1:T}^\phi$ merely consists in optimizing the associated log-likelihood $\mathcal{L}^\phi(\mathbf{x}_{1:T}^\phi, \boldsymbol{\theta}^\phi)$[1]:

$$
\begin{aligned}
\mathcal{L}^\phi(\mathbf{x}_{1:T}^\phi; \boldsymbol{\theta}^\phi) &= \ln \left( P(\mathbf{x}_1^\phi) \prod_{t=2}^{T} P(\mathbf{x}_t^\phi | \mathbf{x}_{t-1}^\phi) \right) \\
&= g(\boldsymbol{\mu}_1^\phi, \Sigma_1^\phi) - \frac{1}{2\sigma^2} \sum_{t=2}^{T} \|\mathbf{x}_t^\phi - A^\phi \mathbf{x}_{t-1}^\phi - \boldsymbol{\mu}^\phi\|^2 - \frac{1}{2} p(T-1) \ln \sigma^2
\end{aligned}
$$

where $p$ is the dimension of $\mathcal{H}$, $g(\boldsymbol{\mu}_1^\phi, \Sigma_1^\phi)$ is a function straightforward to compute which we let aside as it does not add any complexity in setting the gradient of $\mathcal{L}^\phi$ to $\mathbf{0}$. Indeed, performing this task leads to the equations:

$$
\begin{aligned}
A^\phi &= \left( \sum_{t=2}^{T} \mathbf{x}_t^\phi \mathbf{x}_{t-1}^{\phi}{}' - \frac{1}{T-1} \sum_{t=2}^{T} \mathbf{x}_t^\phi \sum_{t=2}^{T} \mathbf{x}_{t-1}^{\phi}{}' \right) \\
&\quad \left( \sum_{t=2}^{T} \mathbf{x}_{t-1}^\phi \mathbf{x}_{t-1}^{\phi}{}' - \frac{1}{T-1} \sum_{t=2}^{T} \mathbf{x}_{t-1}^\phi \sum_{t=2}^{T} \mathbf{x}_{t-1}^{\phi}{}' \right)^{-1}
\end{aligned} \tag{3}
$$

$$
\boldsymbol{\mu}^\phi = \frac{1}{T-1} \sum_{t=2}^{T} \left( \mathbf{x}_t^\phi - A^\phi \mathbf{x}_{t-1}^\phi \right) \tag{4}
$$

$$
\sigma^2 = \frac{1}{p(T-1)} \sum_{t=2}^{T} \|\mathbf{x}_t^\phi - A^\phi \mathbf{x}_{t-1}^\phi - \boldsymbol{\mu}^\phi\|^2 \tag{5}
$$

which require to address two problems: inverting a matrix which could be of infinite dimension (e.g., if a gaussian kernel is used) and/or singular (equation (3)) and making a division by the dimension of the feature space ($p$ in equation (5)).

A general solution to circumvent these problems is to introduce an orthonormal basis $U = \{\mathbf{u}_1^\phi, \ldots, \mathbf{u}_m^\phi\}$ for the subspace $\mathcal{H}_\mathbf{x}$ of $\mathcal{H}$ spanned by $\mathbf{x}_{1:T}^\phi$. For instance, $U$ can be obtained by computing the set of principal components with non-zero eigenvalues of $\mathbf{x}_{1:T}^\phi$ following the procedure proposed in [6]. Once such a set of vectors is available, trying to find good parameters for the model (2) is equivalent to finding an $m$-dimensional linear dynamical model for the sequence $\mathbf{z}_{1:T} = \{\mathbf{z}_1, \ldots, \mathbf{z}_T\}$ where $\mathbf{z}_t$ is the vector of coordinates of $\mathbf{x}_t^\phi$ with respect to $U$, i.e.:

$$\mathbf{z}_t = \left[ \langle \mathbf{x}_t^\phi, \mathbf{u}_1^\phi \rangle \ \langle \mathbf{x}_t^\phi, \mathbf{u}_2^\phi \rangle \cdots \langle \mathbf{x}_t^\phi, \mathbf{u}_m^\phi \rangle \right]' \ \forall t = 1, \ldots, T. \tag{6}$$

Given $\mathbf{z}_{1:T}$, the following linear dynamical model has to be considered:

$$\mathbf{z}_{t+1} = A_z \mathbf{z}_t + \boldsymbol{\mu}_z + \boldsymbol{\nu}_z \tag{7}$$

where $\boldsymbol{\nu}_z$ is again a gaussian noise vector of variance $\sigma^2$. Determining a basis of $\mathcal{H}_\mathbf{x}$ allows to learn the linear dynamical model (7). As it is based on the coordinates of the observed vectors $\mathbf{x}_1^\phi, \ldots, \mathbf{x}_T^\phi$ with respect to the basis, it is equivalent to learning (2). The parameters are estimated thanks to equations (3), (4) and (5) where $\mathbf{x}_t^\phi$ is replaced with $\mathbf{z}_t$ and $p$ with $m$.

For the sake of generalization ability, it might be useful to choose $A_z$ as simple as possible [15]. To do this, we put a penalization on matrices $A_z$ having large values, by imposing a *prior* distribution $p_A$ on $A_z$ defined as: $p_A(A_z) \propto \exp(-\frac{\gamma}{2} \text{trace}\,(A_z' A_z))$, $\gamma > 0$. The computation of the *maximum a posteriori* values for $A$, $\boldsymbol{\mu}$ and $\sigma^2$ is very similar to (3), (4) and (5) except that a few iterations of gradient ascent have to be done.

### 2.3 Back to the Input Space: the Preimage Problem

**The problem** Predicting the future observations with model (7) gives vectors in the feature space $\mathcal{H}$ while vectors from the input space $\mathbb{R}^d$ are needed. Given a vector $\mathbf{z}^\phi$ in $\mathcal{H}$, finding a good vector $\mathbf{x}$ in $\mathbb{R}^d$ such that $\phi(\mathbf{x})$ is as close as possible to $\mathbf{z}^\phi$ is known as the *preimage* problem.

Mika et al. [6] propose to tackle this problem considering the optimization problem:

$$\min_{\mathbf{x}} \|\phi(\mathbf{x}) - \mathbf{z}^\phi\|^2.$$

This problem can be solved efficiently by gradient descent techniques for gaussian kernels. Nevertheless, it may require several optimization phases with different starting points to be ran when other kernels are used (e.g. polynomial kernels of some particular degree).

Here, we propose to use Support Vector Regression (SVR) to solve the preimage problem. This avoids any local minimum problem and allows to benefit from the fact that we have to work with vectors from the inner product space $\mathcal{H}$. In addition, using this strategy, there is no need to solve an optimization problem each time a preimage has to be computed.

**SVR and Preimages Learning** Given a sample dataset $\mathcal{S} = \{(\mathbf{z}_1, y_1), \ldots, (\mathbf{z}_\ell, y_\ell)\}$ with pairs in $\mathcal{Z} \times \mathbb{R}$, the SVR algorithm assumes a structure on $\mathcal{Z}$ given by a kernel $k_z$ and its associated mapping $\phi$ and feature space $\mathcal{H}$ (see [15]). It proceeds as follows (see [14] and [15] for further details). Given a real positive value $\varepsilon$, the algorithm determines a function $f$ such that (a) it maps each $\mathbf{z}_i$ to a value not having deviation larger than $\varepsilon$

from $y_i$, and (b) it is as flat as possible. This function computes its output as $f(\mathbf{z}) = \sum_{i=1}^{\ell}(\alpha_i^* - \alpha_i)k_z(\mathbf{z}_i, \mathbf{z}) + b$ where the vectors $\boldsymbol{\alpha}^*$ and $\boldsymbol{\alpha}$ are the solutions of the problem

$$\max_{\boldsymbol{\alpha}^*,\boldsymbol{\alpha}} \quad -\boldsymbol{\varepsilon}'(\boldsymbol{\alpha}^* + \boldsymbol{\alpha}) + \mathbf{y}'(\boldsymbol{\alpha}^* - \boldsymbol{\alpha}) - \frac{1}{2}((\boldsymbol{\alpha}^* - \boldsymbol{\alpha})K_Z(\boldsymbol{\alpha}^* - \boldsymbol{\alpha}) + \frac{1}{C}(\boldsymbol{\alpha}^{*'}\boldsymbol{\alpha}^* + \boldsymbol{\alpha}'\boldsymbol{\alpha}))$$

$$\text{s.t.} \quad \begin{cases} \mathbf{1}'(\boldsymbol{\alpha}^* - \boldsymbol{\alpha}) = 0 \\ \boldsymbol{\alpha}^* \geq \mathbf{0}, \ \boldsymbol{\alpha} \geq \mathbf{0} \end{cases}$$

The vectors involved in this program are of dimension $\ell$, with $\mathbf{1} = [1 \cdots 1]'$, $\mathbf{0} = [0 \cdots 0]'$, $\boldsymbol{\varepsilon} = [\varepsilon \cdots \varepsilon]'$, $\mathbf{y} = [y_1 \cdots y_\ell]'$ and $K_Z$ is the Gram matrix $K_{Zij} = k_z(\mathbf{z}_i, \mathbf{z}_j)$. Here, $\varepsilon$ is the parameter of the Vapnik's $\varepsilon$-insensitive quadratic loss function and $C$ is a user-defined constant penalizing data points which fail to meet the $\varepsilon$-deviation constraint.

In our case, we are interested in learning the mapping from $\mathcal{H}_\mathbf{x}$ to $\mathbb{R}^d$. In order to learn this mapping, we construct $d$ (the dimension of input space) SVR machines $f_1, \ldots, f_d$. Each $f_i$ is trained to estimate the $i^{th}$ coordinate of the vector $\mathbf{x}_t$ given the coordinates vector $\mathbf{z}_t$ of $\mathbf{x}_t$ with respect to $U$. Denoting by $z_u$ the function which maps a vector $\mathbf{x}$ to its coordinate vector $\mathbf{z}$ in $U$, the $d$ machines provide the mapping $\psi$:

$$\begin{array}{rcl} \psi : \mathcal{H}_\mathbf{x} & \rightarrow & \mathbb{R}^d \\ \mathbf{x} & \mapsto & [f_1(z_u(\mathbf{x})) \cdots f_d(z_u(\mathbf{x}))]' \end{array} \tag{8}$$

which can be used to estimate the preimages. Using $\psi$, and noting that the program involved by the SVR algorithm is convex, the estimation of the preimages does not have to deal with any problem of local minima.

# 3    Numerical Results

In this section we present experiments on highly nonlinear time series prediction with Kernel Dynamical Modeling. As the two series we consider are one dimensional we use the following setup. Each series of length $T$ is referred to as $x_{1:T}$. In order to model it, we introduce an embedding dimension $d$ and a step size $\kappa$ such that vectors $\mathbf{x}_t = (x_t, x_{t-\kappa}, \ldots, x_{t-(d-1)\kappa})'$ are considered. We compare the perfomances of KDM to the performances achieved by an SVR for nonlinear time series analysis [7, 8], where the mapping associating $\mathbf{x}_t$ to $x_{t+\kappa}$ is learned. The hyperparameters (kernel parameter and SVR penalization constant $C$) are computed with respect to the one-step prediction error measured on a test set, while the value of $\varepsilon$ is set to 1e-4. Prediction quality is assessed on an independent validation sequence on which root mean squared error (RMSE) is computed.

Two kinds of prediction capacity are evaluated. The first one is a one-step prediction when after a prediction has been made, the true value is used to estimate the next time series output. The second one is a multi-step or trajectory prediction, where the prediction made by a model serves as a basis for the future predictions.

In order to make a prediction for a time $t > T$, we suppose that we are provided with the vector $\mathbf{x}_{t-1}$, which may have been observed or computed. We determine the coordinates $\mathbf{z}_{t-1}$ of $\mathbf{x}_{t-1}^\phi$ with respect to $U$ and infer the value of $\mathbf{z}_t$ by $\mathbf{z}_t = A_z\mathbf{z}_{t-1} + \mu_z$ (see equation (7)); $\psi$ is then used to recover an estimation of $\mathbf{x}_{t+1}$ (cf. equation (8)). In all our experiments we have made the crude –yet efficient– choice of the linear kernel for $k_z$.

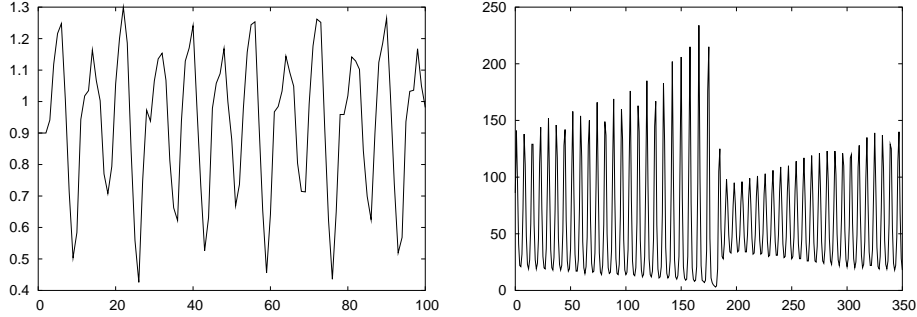

Figure 1: (left) 100 points of the Mackey-Glass time series $MG_{17}$, (right) the first 350 points of the Laser time series.

Table 1: Error (RMSE) of one-step and trajectory predictions for gaussian and polynomial kernels for the time series $MG_{17}$. The regularizing values used for KDM are in subscript. The best results are italicized.

|  | Gaussian | | Polynomial | |
| --- | --- | --- | --- | --- |
| Algo. | 1S | 100S | 1S | 100S |
| SVR | *0.0812* | 0.2361 | 0.1156 | - |
| $KDM_0$ | 0.0864 | 0.2906 | 0.1112 | 0.2975 |
| $KDM_{0.1}$ | 0.0863 | 0.2893 | 0.1112 | 0.2775 |
| $KDM_1$ | 0.0859 | 0.2871 | 0.1117 | 0.2956 |
| $KDM_{10}$ | 0.0844 | 0.2140 | 0.1203 | 0.1964 |
| $KDM_{100}$ | 0.0899 | *0.1733* | *0.0970* | *0.1744* |

### 3.1 Mackey-Glass Time Series Prediction

The Mackey-Glass time series comes from the modeling of blood cells production evolution. It is a one-dimensional signal determined by

$$\frac{dx(t)}{dt} = -0.1x(t) + \frac{0.2x(t-\tau)}{1 + x(t-\tau)^{10}}$$

which, for values of $\tau$ greater than 16.8, shows some highly nonlinear chaotic behavior (see Figure 1 left).

We focus on $MG_{17}$, for which $\tau = 17$, and construct embedding vectors of size $d = 6$ and step size $\kappa = 6$. As $\mathbf{x}_t$ is used to predict $x_{t+\kappa}$, the whole dataset can be divided into six "independent" datasets, the first one $\mathcal{S}_1$ containing $\mathbf{x}_{1+(d-1)\kappa}$, the second one $\mathcal{S}_2$, $\mathbf{x}_{2+(d-1)\kappa}$, ..., and the sixth one $\mathcal{S}_6$, $\mathbf{x}_{d\kappa}$. Learning is done as follows. The first 100 points of $\mathcal{S}_1$ are used to learning, while the first 100 points of $\mathcal{S}_2$ serve to choose the hyperparameters. The prediction error is measured with respect to the points in the range 201 to 300 of $\mathcal{S}_1$.

Table 1 reports the RMSE error obtained with gaussian and polynomial kernels, where 1S and 100S respectively stand for one-step prediction and multi-step prediction over the 100 future observations.

SVR one-step prediction with gaussian kernel gives the best RMSE. None of the tested regularizers allows KDM to perform better, even if the prediction error obtained with them is never more than 10% away from SVR error.

Table 2: Error (RMSE) of one-step and trajectory predictions for gaussian and polynomial kernels for the time series Laser. The regularizing values used for KDM are in subscript.

| Algo. | Gaussian | | Polynomial | |
|---|---|---|---|---|
| | 1S | 100S | 1S | 100S |
| SVR | 15.81 | 67.57 | 18.14 | 66.73 |
| $KDM_0$ | 67.95 | 416.2 | 43.92 | 68.90 |
| $KDM_{0.1}$ | 16.59 | 69.65 | 22.37 | 69.60 |
| $KDM_1$ | *13.96* | 70.16 | 18.13 | 70.65 |
| $KDM_{10}$ | 15.18 | 66.82 | *17.39* | 69.43 |
| $KDM_{100}$ | 18.65 | *56.53* | 17.61 | *53.84* |

KDM trajectory prediction with gaussian kernel and regularizer $\gamma = 100$ leads to the best error. It is around 17% lower than that of SVR multi-step prediction while KDM with no regularizer gives the poorest prediction, emphasizing the importance of the regularizer.

Regarding one-step prediction with polynomial kernel, there is no significant difference between the performance achieved by SVR and that of KDM, when regularizer is 0, 0.1, 1 or 10. For a regularizer $\gamma = 100$, KDM however leads to the best one-step prediction error, around 16% lower than that obtained by SVR prediction.

The dash '-' appearing in the first line of the table means that the trajectory prediction made by the SVR with a polynomial kernel has failed to give finite predictions. On the contrary, KDM never shows this kind of behavior. For a regularizer value of $\gamma = 100$, it even gives the best trajectory prediction error.

### 3.2 Laser Time Series Prediction

The Laser time series is the dataset A from the Santa Fe competition. It is a univariate time series from an experiment conducted in a physics laboratory (Figure 1 (right) represents the first 350 points of the series). An embedding dimension $d = 3$ and a step size $\kappa = 1$ are used. The dataset is divided as follows. The first 100 points are used for training, whereas the points in the range 201 to 300 provide a test set to select hyperparameters. The validation error (RMSE) is evaluated on the points in the range 101 to 200.

Table 2 reports the validation errors obtained for the two kinds of prediction. The most striking information provided by this table is the large error archieved by KDM with no regularizer when a gaussian kernel is used. Looking at the other RMSE values corresponding to different regularizers, the importance of penalizing transition matrices with large entries is underlined.

Besides, when the regularizer $\gamma$ is appropriately chosen, we see that KDM with a gaussian kernel can achieve very good predictions, for the one-step prediction and the multi-step prediction as well. KDM one-step best prediction error is however not as far from SVR one-step prediction (about 10% lower) than KDM multi-step is from its SVR counterpart (around 16% lower).

When a polynomial kernel is used, we observe that KDM with no regularizer provides poor results with regards to the one-step prediction error. Contrary to what occurs with the use of a gaussian kernel, KDM with no regularization does not show bad multi-step prediction ability. Looking at the other entries of this table once again shows that KDM can give very good predictions when a well-suited regularizer is chosen. Hence, we notice that the best multi-step prediction error of KDM is above 19% better than that obtained by SVR

multi-step prediction.

## 4  Discussion

### 4.1  Another Way of Choosing the Parameters

The introduction of a basis $U$ allows to find the parameters of KDM without computing any inversion of infinite dimensional matrices or division by the dimension of $\mathcal{H}$. There is, however a more elegant way to find these parameters when $\sigma^2$ is assumed to be known. In this case, equation (5) needs not to be considered any longer. Considering the prior $p_A(A^\phi) \propto \exp(-\frac{\gamma}{2\sigma^2} \text{trace } (A^{\phi'} A^\phi))$, for a user defined $\gamma$, the *maximum a posteriori* for $A^\phi$ is obtained as:

$$A^\phi = \left( \sum_{t=2}^{T} \mathbf{x}_t^\phi \mathbf{x}_{t-1}^{\phi}{}' - \frac{1}{T-1} \sum_{t=2}^{T} \mathbf{x}_t^\phi \sum_{t=2}^{T} \mathbf{x}_{t-1}^{\phi}{}' \right)$$
$$\left( \gamma I + \sum_{t=2}^{T} \mathbf{x}_{t-1}^\phi \mathbf{x}_{t-1}^{\phi}{}' - \frac{1}{T-1} \sum_{t=2}^{T} \mathbf{x}_{t-1}^\phi \sum_{t=2}^{T} \mathbf{x}_{t-1}^{\phi}{}' \right)^{-1}.$$

Introducing the matrix $X^\phi = [\mathbf{x}_1^\phi \cdots \mathbf{x}_T^\phi]$, the $T$-dimensional vectors $\mathbf{f} := [0 \ 1 \cdots 1]'$, $\mathbf{g} := [1 \cdots 1 \ 0]$, the $T \times T$ matrix $P = (P_{ij}) = (\delta_{i,j+1})$ defining $J = P - \mathbf{f}\,\mathbf{g}/(T-1)'$ and $M = \text{diag}\,(\mathbf{g}) - \mathbf{g}\mathbf{g}'/(T-1)$, $A^\phi$ can be rewritten as

$$A^\phi = \left( X^\phi J X^{\phi'} \right) \left( \gamma I + X^\phi M X^{\phi'} \right)^{-1}$$
$$= \frac{1}{\gamma} X^\phi J \left[ I - \frac{1}{\gamma} K M (I + \frac{1}{\gamma} M K M)^{-1} M \right] X^{\phi'}$$

thanks to the Sherman-Woodbury formula, $K$ being the Gram matrix associated to $\mathbf{x}_{1:T}^\phi$. It is thus possible to directly determine the matrix $A^\phi$ when $\sigma^2$ is known, the same holding for $\boldsymbol{\mu}^\phi$ since equation (5) remains unchanged.

### 4.2  Link to Kalman Filtering

The usual way to recover a noisy nonlinear signal is to use the Extended Kalman Filter (EKF) or the Unscented Kalman Filter (UKF) [4]. The use of these algorithms involves two steps. First, the clean dynamics, as given by $h$ in equation (1) is learned by a regressor, e.g., a multilayer perceptron. Given a noisy time series from the same driving process $h$, EKF and UKF then process that series by respectively a first-order linearization of $h$ and an efficient 'sampling' method to determine the clean signal. Apart from the latter essential approximations done by these algorithms, the core of EKF and UKF resembles that of classical Kalman filtering (and smoothing).

Regarding the performances of KDM to learn a complex dynamics, it could be directly used to model the process $h$. In addition, its matricial formulation is suitable to the traditional matrices computations involved by the filtering task (see [5, 11] for details). Hence, a link between KDM and Kalman filtering has been the purpose of [9, 10] where a nonlinear Kalman filter based on the use of kernels is proposed: the ability of the proposed model to address the modeling of nonlinear dynamics is demonstrated, while the classical procedures (even the EM algorithm) associated to linear dynamical systems remain valid.

## 5  Conclusion and Future Work

Three main results are presented: first, we introduce KDM, a kernel extension of linear dynamical models and show how the kernel trick allows to learn a linear model in a feature

space associated to a kernel. Second, an original and efficient solution based on learning has been applied for the preimage problem. Third, Kernel Dynamical Model can be linked to the Kalman filter model with a hidden state process living in the feature space.

In the framework of time series prediction, KDM proves to work very well and to compete with the best time series predictors particularly on long time range prediction.

To conclude, this work can lead to several future directions. All classic tasks involving a dynamic setting such as filtering/predicting (e.g., tracking) and smoothing (e.g., time series denoising) can be tackled by our approach and have to be tested. As pointed out by [9, 10], the kernel approach can also be applied to linear dynamical models with hidden states to provide a kernelized version of the Kalman filter, particularly allowing the implementation of an exact nonlinear EM procedure (involving closed form equations as the method proposed by [3]). Besides, the use of kernel opens the door to dealing with structured data, making KDM a very attractive tool in many areas such as bioinformatics, texts and video application. Lastly, from the theoretical point of view, a very interesting issue is that of the actual noise corresponding to a gaussian noise in a feature space.

## Footnotes

[1] $\boldsymbol{\theta}^\phi := \{A^\phi, \boldsymbol{\mu}^\phi, \sigma^2, \boldsymbol{\mu}_1^\phi, \Sigma_1^\phi\}$, and $\boldsymbol{\mu}_1^\phi$ and $\Sigma_1^\phi$ are the parameters of the gaussian vector $\mathbf{x}_1^\phi$.

## References

[1] B. Boser, I. Guyon, and V. Vapnik. A Training Algorithm for Optimal Margin Classifiers. In *Proc. of the 5th Annual Workshop on Comp. Learning Theory*, volume 5, 1992.

[2] G. Dorffner. Neural networks for time series processing. *Neural Network World*, 6(4):447–468, 1996.

[3] Z. Ghahramani and S. Roweis. Learning nonlinear dynamical systems using an em algorithm. In M. S. Kearns, S. A. Solla, and D. A. Cohn, editors, *Advances in Neural Information Processing Systems*, volume 11, pages 599–605. MIT Press, 1999.

[4] S. Julier and J. Uhlmann. A New Extension of the Kalman Filter to Nonlinear Systems. In *Int. Symp. Aerospace/Defense Sensing, Simul. and Controls*, 1997.

[5] R. E. Kalman. A New Approach to Linear Filtering and Prediction Problems. *Transactions of the ASME–Journal of Basic Engineering*, 82(Series D):35–45, 1960.

[6] S. Mika, B. Schölkopf, A. J. Smola, K.-R. Müller, M. Scholz, and G. Rätsch. Kernel PCA and De-Noising in Feature Spaces. In *NIPS*. MIT Press, 1999.

[7] S. Mukherjee, E. Osuna, and F. Girosi. Nonlinear prediction of chaotic time series using support vector machines. In *Proc. of IEEE NNSP'97*, 1997.

[8] K. Müller, A. Smola, G. Rätsch, B. Schölkopf, J. Kohlmorgen, and V. Vapnik. Predicting Time Series with Support Vector Machines. In W. Gerstner, A. Germond, M. Hasler, and J.-D. Nicoud, editors, *Artificial Neural Networks - ICANN'97*, pages 999–1004. Springer, 1997.

[9] L. Ralaivola. *Modélisation et apprentissage de concepts et de systèmes dynamiques*. PhD thesis, Université Paris 6, France, 2003.

[10] L. Ralaivola and F. d'Alché-Buc. Filtrage de Kalman non linéaire à l'aide de noyaux. In *Actes du $19^{eme}$ Symposium GRETSI sur le traitement du signal et des images*, 2003.

[11] A-V.I. Rosti and M.J.F. Gales. Generalised linear Gaussian models. Technical Report CUED/F-INFENG/TR.420, Cambridge University Engineering Department, 2001.

[12] S. Roweis and Z. Ghahramani. A unifying review of linear Gaussian models. *Neural Computation*, 11(2):305–345, 1997.

[13] B. Schölkopf and A. J. Smola. *Learning with Kernels, Support Vector Machines, Regularization, Optimization and Beyond*. MIT University Press, 2002.

[14] A. Smola and B. Schölkopf. A Tutorial on Support Vector Regression. Technical Report NC2-TR-1998-030, NeuroCOLT2, 1998.

[15] V. Vapnik. *Statistical Learning Theory*. John Wiley and Sons, inc., 1998.
